# A Bayesian Model of Conditioned Perception

**Alan A. Stocker**[*] and **Eero P. Simoncelli**

Howard Hughes Medical Institute,
Center for Neural Science,
and Courant Institute of Mathematical Sciences
New York University
New York, NY-10003, U.S.A.

We argue that in many circumstances, human observers evaluate sensory evidence simultaneously under multiple hypotheses regarding the physical process that has generated the sensory information. In such situations, inference can be optimal if an observer combines the evaluation results under each hypothesis according to the probability that the associated hypothesis is correct. However, a number of experimental results reveal suboptimal behavior and may be explained by assuming that once an observer has committed to a particular hypothesis, subsequent evaluation is based on that hypothesis alone. That is, observers sacrifice optimality in order to ensure *self-consistency*. We formulate this behavior using a conditional Bayesian observer model, and demonstrate that it can account for psychophysical data from a recently reported perceptual experiment in which strong biases in perceptual estimates arise as a consequence of a preceding decision. Not only does the model provide quantitative predictions of subjective responses in variants of the original experiment, but it also appears to be consistent with human responses to cognitive dissonance.

## 1 Motivation

Is the glass half full or half empty? In different situations, the very same perceptual evidence (*e.g.* the perceived level of liquid in a glass) can be interpreted very differently. Our perception is *conditioned* on the context within which we judge the evidence. Perhaps we witnessed the process of the glass being filled, and thus would more naturally think of it as half full. Maybe it is the only glass on the table that has liquid remaining, and thus its precious content would be regarded as half full. Or maybe we simply like the content so much that we cannot have enough, in which case we may view it as being half empty.

Contextual influences in low-level human perception are the norm rather than the exception, and have been widely reported. Perceptual illusions, for example, often exhibit particularly strong contextual effects, either in terms of perceptual space (*e.g.* spatial context affects perceived brightness; see [1] for impressive examples) or time (prolonged exposure to an adaptor stimulus will affect subsequent perception, see *e.g.* the motion after-effect [2]). Data of recent psychophysical experiments suggest that an observer's previous perceptual decisions provide additional form of context that can substantially influence subsequent perception [3, 4]. In particular, the outcome of a categorical decision task can strongly bias a subsequent estimation task that is based on the same stimulus presentation. Contextual influences are typically strongest when the sensory evidence is most ambiguous in terms of its interpretation, as in the example of the half-full (or half-empty) glass.

Bayesian estimators have proven successful in modeling human behavior in a wide variety of low-level perceptual tasks (for example: cue-integration (see *e.g.* [5]), color perception (*e.g.* [6]), visual motion estimation (*e.g.* [7, 8])). But they generally do not incorporate contextual dependencies

---

[*]corresponding author.

beyond a prior distribution (reflecting past experience) over the variable of interest. Contextual dependencies may be incorporated in a Bayesian framework by assuming that human observers, when performing a perceptual task, test different hypotheses about the underlying structure of the sensory evidence, and arrive at an estimate by weighting the estimates under each hypothesis according to the strength of their belief in that hypothesis. This approach is known as optimal model evaluation [9], or Bayesian model averaging [10] and has been previously suggested to account for cognitive reasoning [11]. It further has been suggested that the brain could use different neuro-modulators to keep track of the probabilities of individual hypotheses [12]. Contextual effects are reflected in the observer's selection and evaluation of these hypotheses, and thus vary with experimental conditions. For the particular case of cue-integration, Bayesian model averaging has been proposed and tested against data [13, 14], suggesting that some of the observed non-linearities in cue integration are the result of the human perceptual system taking into account multiple potential contextual dependencies.

In contrast to these studies, however, we propose that model averaging behavior is abandoned once the observer has committed to a particular hypothesis. Specifically, subsequent perception is conditioned only on the chosen hypothesis, thus sacrificing optimality in order to achieve *self-consistency*. We examine this hypothesis in the context of a recent experiment in which subjects were asked to estimate the direction of motion of random dot patterns after being forced to make a categorical decision about whether the direction of motion fell on one side or the other of a reference mark [4]. Depending on the different levels of motion coherence, responses on the estimation task were heavily biased by the categorical decision. We demonstrate that a self-consistent conditional Bayesian model can account for mean behavior, as well as behavior on individual trials [8]. The model has essentially no free parameters, and in addition is able to make precise predictions under a wide variety of alternative experimental arrangements. We provide two such example predictions.

## 2   Observer Model

We define perception as a statistical estimation problem in which an observer tries to infer the value of some environmental variable $s$ based on sensory evidence $m$ (see Fig. 1). Typically, there are sources of uncertainty associated with $m$, including both sensor noise and uncertainty about the relationship between the sensory evidence and the variable $s$. We refer to the latter as *structural uncertainty* which represents the degree of ambiguity in the observer's interpretation of the physical world. In cases where the structural possibilities are discrete, we denote them as a set of hypotheses $H = \{h_1, ..., h_N\}$. Perceptual inference requires two steps. First, the observer computes their belief

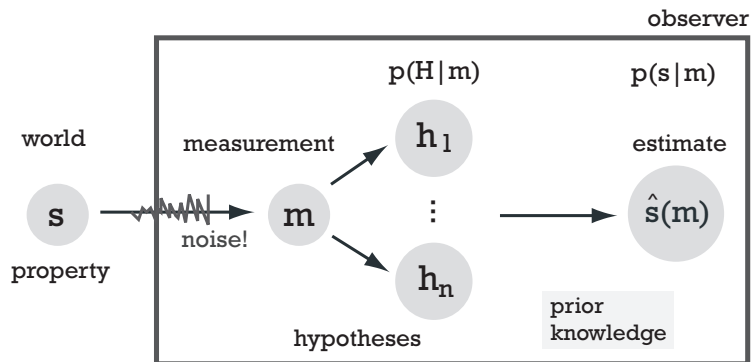

Figure 1: *Perception as conditioned inference problem.* Based on noisy sensory measurements $m$ the observer generates different hypotheses for the generative structure that relates $m$ to the stimulus variable $s$. Perception is a two-fold inference problem: Given the measurement and prior knowledge, the observer generates and evaluates different structural hypotheses $h_i$. Conditioned on this evaluation, they then infer an estimate $\hat{s}(m)$ from the measurement $m$.

in each hypothesis for given sensory evidence $m$. Using Bayes' identity, the belief is expressed as

the posterior

$$p(H|m) = \frac{p(m|H)p(H)}{p(m)} \ . \tag{1}$$

Second, for each hypothesis, a conditional posterior is formulated as $p(s|m, H = h_i)$, and the full (non-conditional) posterior is computed by integrating the evidence over all hypotheses, weighted by the belief in each hypothesis $h_i$:

$$p(s|m) = \sum_{i=1}^{N} p(s|m, H = h_i)p(H = h_i|m) \ . \tag{2}$$

Finally, the observer selects an estimate $\hat{s}$ that minimizes the expected value (under the posterior) of an appropriate loss function [1].

## 2.1 Decision leads to conditional estimation

In situations where the observer has already made a decision (either explicit or implicit) to select one hypothesis as being correct, we postulate that subsequent inference will be based on that hypothesis alone, rather than averaging over the full set of hypotheses. For example, suppose the observer selects the *maximum a posteriori* hypothesis $h_{\text{MAP}}$, the hypothesis that is most probable given the sensory evidence and the prior distribution. We assume that this decision then causes the observer to reset the posterior probabilities over the hypotheses to

$$\begin{aligned} p(H|m) &= 1, && \text{if } H = h_{\text{MAP}} \\ &= 0, && \text{otherwise.} \end{aligned} \tag{3}$$

That is, the decision making process forces the observer to consider the selected hypothesis as correct, with all other hypotheses rendered impossible. Changing the beliefs over the hypotheses will obviously affect the estimate $\hat{s}$ in our model. Applying the new posterior probabilities Eq. (3) simplifies the inference problem Eq. (2) to

$$p(s|m) = p(s|m, H = h_{\text{MAP}}) \ . \tag{4}$$

We argue that this *simplification by decision* is essential for complex perceptual tasks (see Discussion). By making a decision, the observer frees resources, eliminating the need to continuously represent probabilities about other hypotheses, and also simplifies the inference problem. The price to pay is that the subsequent estimate is typically biased and sub-optimal.

## 3 Example: Conditioned Perception of Visual Motion

We tested our observer model by simulating a recently reported psychophysical experiment [4]. Subjects in this experiment were asked on each trial to *decide* whether the overall motion direction of a random dot pattern was to the right or to the left of a reference mark (as seen from the fixation point). Low levels of motion coherence made the decision task difficult for motion directions close to the reference mark. In a subset of randomly selected trials subjects were also asked to *estimate* the precise angle of motion direction (see Fig. 2). The decision task was always preceding the estimation task, but at the time of the decision, subjects were unaware whether they would had to perform the estimation task or not.

## 3.1 Formulating the observer model

We denote $\theta$ as the direction of coherent motion of the random dot pattern, and $m$ the noisy sensory measurement. Suppose that on a given trial the measurement $m$ indicates a direction of motion to the right of the reference mark. The observer can consider two hypotheses $H = \{h_1, h_2\}$ about the actual physical motion of the random dot pattern: Either the true motion is actually to the right and thus in agreement with the measurement, or it is to the left but noise has disturbed the measurement

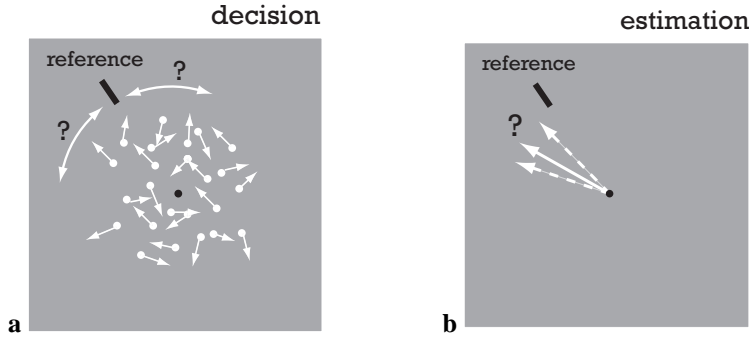

Figure 2: *Decision-estimation experiment.* (a) Jazayeri and Movshon presented moving random dot patterns to subjects and asked them to decide if the overall motion direction was either to the right or the left of a reference mark [4]. Random dot patterns could exhibit three different levels of motion coherence (3, 6, and 12%) and the single coherent motion direction was randomly selected from a uniform distribution over a symmetric range of angles $[-\alpha, \alpha]$ around the reference mark. (b) In randomly selected 30% of trials, subjects were also asked, after making the directional decision, to estimate the exact angle of motion direction by adjusting an arrow to point in the direction of perceived motion. In a second version of the experiment, motion was either toward the direction of the reference mark or in the opposite direction.

such that it indicates motion to the right. The observer's belief in each of the two hypotheses based on their measurement is given by the posterior distribution according to Eq. (1), and the likelihood

$$p(m|H) = \int_{-\pi}^{\pi} p(m|\theta, H)p(\theta|H)d\theta \ . \tag{5}$$

The optimal decision is to select the hypothesis $h_{\text{MAP}}$ that maximizes the posterior given by Eq. (1).

## 3.2 Model observer vs. human observer

The subsequent *conditioned estimate* of motion direction then follows from Eq. (4) which can be rewritten as

$$p(\theta|m) = \frac{p(m|\theta, H = h_{\text{MAP}})p(\theta|H = h_{\text{MAP}})}{p(m|H = h_{\text{MAP}})} \ . \tag{6}$$

The model is completely characterized by three quantities: The likelihood functions $p(m|\theta, H)$, the prior distributions $p(\theta|H)$ of the direction of motion given each hypothesis, and the prior on the hypotheses $p(H)$ itself (shown in Fig. 3). In the given experimental setup, both prior distributions were uniform but the width parameter of the motion direction $\alpha$ was not explicitly available to the subjects and had to be individually learned from training trials. In general, subjects seem to over-estimate this parameter (up to a factor of two), and adjusting its value in the model accounts for most of the variability between subjects. The likelihood functions $p(m|\theta, H)$ is given by the uncertainty about the motion direction due to the low motion coherence levels in the stimuli and the sensory noise characteristics of the observer. We assumed it to be Gaussian with a width that varies inversely with the coherence level. Values were estimated from the data plots in [4].

Figure 4 compares the prediction of the observer model with human data. Trial data of the model were generated by first sampling a hypothesis $h'$ according to $p(H)$, then drawing a stimulus direction from $p(\theta|H = h')$. then picking a sensory measurement sample $m$ according to the conditional probability $p(m|\theta, H = h')$, and finally performing inference according to Eqs. (1) and (6). The model captures the characteristics of human behavior in both the decision and the subsequent estimation task. Note the strong influence of the decision task on the subsequent estimation of the motion direction, effectively pushing the estimates away from the decision boundary.

We also compared the model with a second version of the experiment, in which the decision task was to discriminate between motion toward and away from the reference [4]. Coherent motion of the random dot pattern was uniformly sampled from a range around the reference and from a range

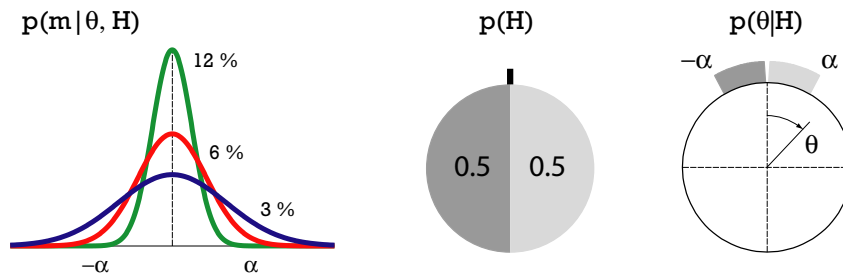

Figure 3: *Ingredients of the conditional observer model.*  The sensory signal is assumed to be corrupted by additive Gaussian noise, with width that varies inversely with the level of motion coherence. Actual widths were approximated from those reported in [4]. The prior distribution over the hypotheses $p(H)$ is uniform. The two prior distributions over motion direction given each hypothesis, $p(\theta|H = h_{1,2})$, are again determined by the experimental setup, and are uniform over the range $[0, \pm\alpha]$.

around the direction opposite to the reference, as illustrated by the prior distributions shown in Fig. 5. Again, note that these distributions are given by the experiment and thus, assuming the same noise characteristics as in the first experiment, the model has no free parameters.

## 3.3  Predictions

The model framework also allows us to make quantitative predictions of human perceptual behavior under conditions not yet tested. Figure 6 shows the model observer's behavior under two modifications of the original experiment. The first is identical to the experiment shown in Fig. 4 but with unequal prior probability on the two hypotheses. The model predicts that a human subject would respond to this change by more frequently choosing the more likely hypothesis. However, this hypothesis would also be more likely to be correct, and thus the estimates under this hypothesis would exhibit less bias than in the original experiment.

The second modification is to add a second reference and ask the subject to decide between three different classes of motion direction (*e.g.* left, central, right). Again, the model predicts that in such a case, a human subject's estimate in the central direction should be biased away from both decision boundaries, thus leading to an almost constant direction estimate. Estimates following a decision in favor of the two outer classes show the same repulsive bias as seen in the original experiment.

## 4  Discussion

We have presented a normative model for human perception that captures the conditioning effects of decisions on an observer's subsequent evaluation of sensory evidence. The model is based on the premise that observers aim for optimal inference (taking into account all sensory evidence and prior information), but that they exhibit decision-induced biases because they also aim to be self-consistent, eliminating alternatives that have been decided against. We've demonstrated that this model can account for the experimental results of [4].

Although this strategy is suboptimal (in that it does not minimize expected loss), it provides two fundamental advantages. First, self-consistency would seem an important requirement for a stable interpretation of the environment, and adhering to it might outweigh the disadvantages of perceptual misjudgments. Second, framing perception in terms of optimal statistical estimation implies that the more information an observer evaluates, the more accurately they should be able to solve a perceptual task. But this assumes that the observer can construct and retain full probability distributions and perform optimal inference calculations on these. Presumably, accumulating more probabilistic evidence of more complex conditional dependencies has a cost, both in terms of storage, and in terms of the computational load of performing subsequent inference. Thus, discarding information after making a decision can help to keep this storage and the computational complexity at a manageable level, freeing computational resources to perform other tasks.

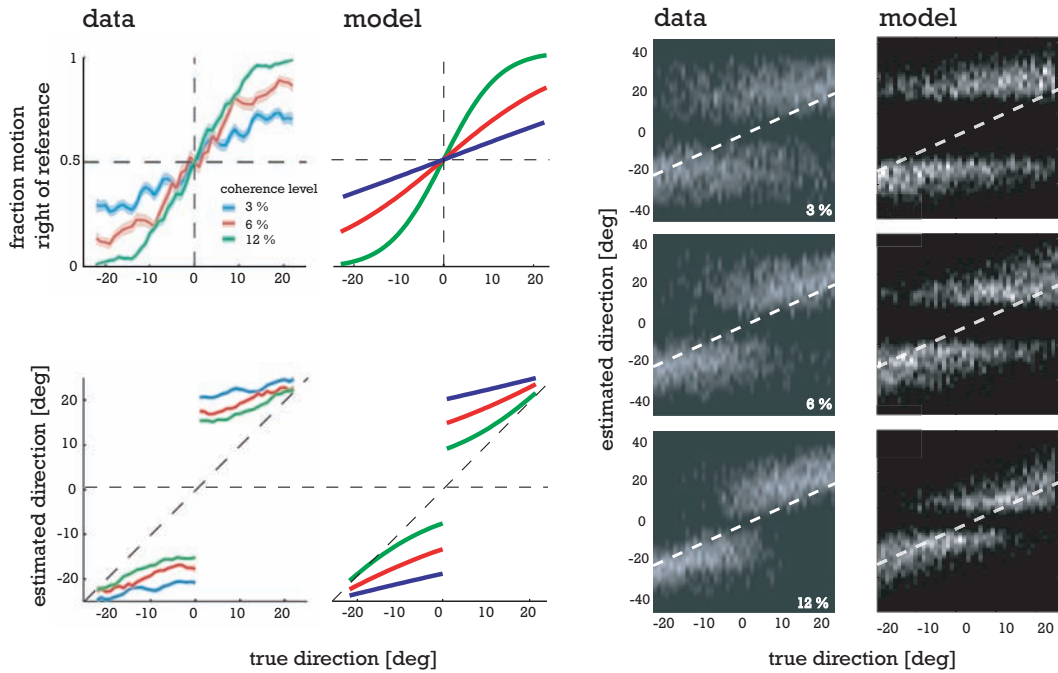

Figure 4: *Comparison of model predictions with data for a single subject.* Upper left: The two panels show the percentage of observed motion to the right as a function of the true pattern direction, for the three coherence levels tested. The model accurately predicts the subject's behavior, which exhibits a decrease in the number of false decisions with decreasing noise levels and increasing distance to the reference. Lower left: Mean estimates of the direction of motion after performing the decision task. Clearly, the decision has a substantial impact on the subsequent estimate, producing a strong bias away from the reference. The model response exhibits biases similar to those of the human subjects, with lower coherence levels producing stronger repulsive effects. Right: Grayscale images show distributions of estimates across trials for both the human subject and the model observer, for all three coherence levels. All trials are included (correct and incorrect). White dashed lines represent veridical estimates. Model observer performed 40 trials at each motion direction (in 1.5 degrees increments). Human data are replotted from [4].

An interesting avenue for exploration is the implementation of such an algorithm in neural substrate. Recent studies propose a means by which population of neurons can represent and multiply probability distributions [15]. It would be worthwhile to consider how the model presented here could be implemented with such a neural mechanism. In particular, one might expect that the sudden change in posterior probabilities over the hypotheses associated with the decision task would be reflected in sudden changes in response pattern in such populations [16].

Questions remain. For the experiment we have modeled, the hypotheses were specified by the two alternatives of the decision task, and the subjects were forced to choose one of them. What happens in more general situations? First, do humans *always* decompose perceptual inference tasks into a set of inference problems, each conditioned on a different hypothesis? Data from other, cue-combination experiments suggest that subjects indeed seem to perform such probabilistic decomposition [13, 14]. If so, then how do observers generate these hypotheses? In the absence of explicit instructions, humans may automatically perform implicit comparisons relative to reference features that are unconsciously selected from the environment. Second, if humans do consider different hypotheses, do they always select a single one on which subsequent percepts are conditioned, even if not explicitly asked to do so? For example, simply displaying the reference mark in the experiment of [4] (without asking the observer to report any decision) might be sufficient to trigger an implicit decision that would result in behaviors similar to those shown in the explicit case.

Finally, although we have only tested it on data of a particular psychophysical experiment, we believe that our model may have implications beyond low-level sensory perception. For instance, a

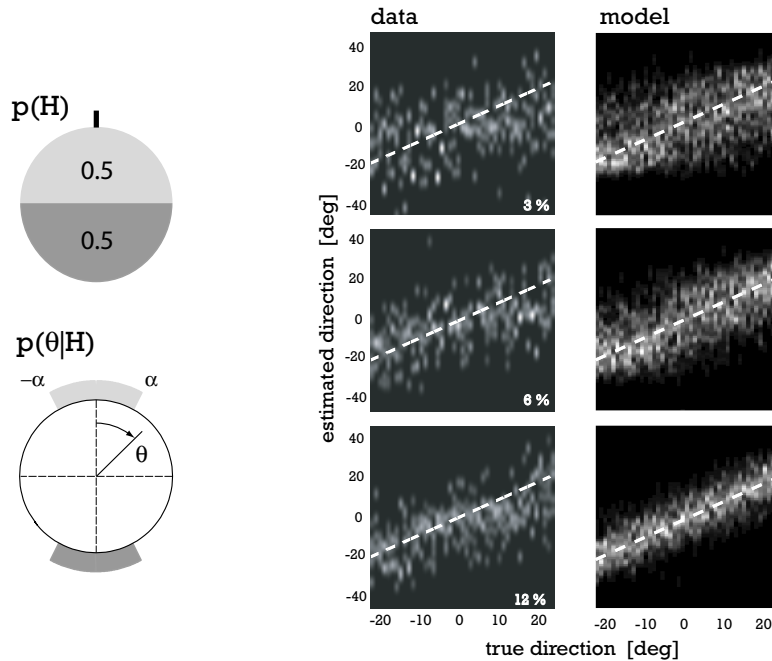

Figure 5: *Comparison of model predictions with data for second experiment.* Left: Prior distributions for second experiment in [4]. Right: Grayscale images show the trial distributions of the human subject and the model observer for all three coherence levels. White dashed lines represent veridical estimates. Note that the human subject does not show any significant bias in their estimate. The trial variance appears to increase with decreasing levels of coherence. Both characteristics are well predicted by the model. Human data replotted from [4] (supplementary material).

well-studied human attribute is known as *cognitive dissonance* [17], which causes people to adjust their opinions and beliefs to be consistent with their previous statements or behaviors. [2] Thus, self-consistency may be a principle that governs computations throughout the brain.

## Acknowledgments

We thank J. Tenenbaum for referring us to the cognitive dissonance literature, and J. Pillow, N. Daw, D. Heeger, A. Movshon, and M. Jazayeri for interesting discussions.

## Footnotes

[1]For the purpose of this paper, we assume a standard squared error loss function, in which case the observer should choose the mean of the posterior distribution.

[2]An example that is directly analogous to the perceptual experiment in [4] is documented in [18]: Subjects initially rated kitchen appliances for attractiveness, and then were allowed to select one as a gift from amongst two that they had rated equally. They were subsequently asked to rate the appliances again. The data show a repulsive bias of the post-decision ratings compared with the pre-decision ratings, such that the rating of the selected appliance increased, and the rating of the rejected appliance decreased.

## References

[1] E.H. Adelson. Perceptual organization and the judgment of brightness. *Science*, 262:2042–2044, December 1993.

[2] S.P. Thompson. Optical illusions of motion. *Brain*, 3:289–298, 1880.

[3] S. Baldassi, N. Megna, and D.C. Burr. Visual clutter causes high-magnitude errors. *PLoS Biology*, 4(3):387ff, March 2006.

[4] M. Jazayeri and J.A. Movshon. A new perceptual illusion reveals mechanisms of sensory decoding. *Nature*, 446:912ff, April 2007.

[5] M.O. Ernst and M.S. Banks. Humans integrate visual and haptic information in a statistically optimal fashion. *Nature*, 415:429ff, January 2002.

[6] D. Brainard and W. Freeman. Bayesian color constancy. *Journal of Optical Society of America A*, 14(7):1393–1411, July 1997.

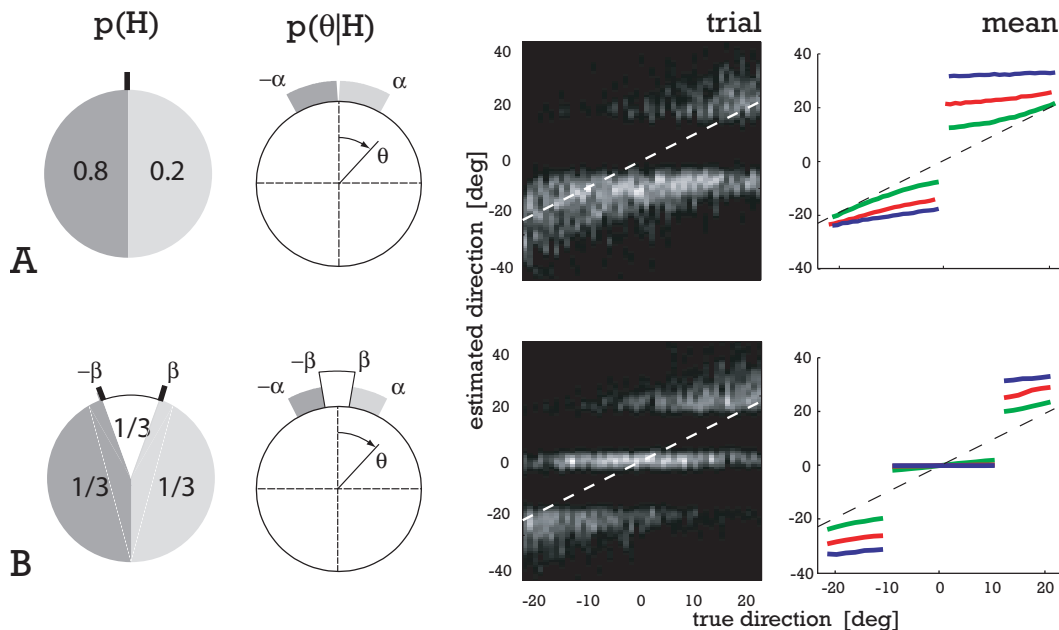

Figure 6: *Model predictions for two modifications of the original experiment.* A: We change the prior probability $p(H)$ to be asymmetric (0.8 vs. 0.2). However, we keep the prior distribution of motion directions given a particular side $p(\theta|H)$ constant within the range $[0, \pm\alpha]$. The model makes two predictions (trials shown for an intermediate coherence level): First, although tested with an equal number of trials for each motion direction, there is a strong bias induced by the asymmetric prior. And second, the direction estimates on the left are more veridical than on the right. B: We present *two* reference marks instead of one, asking the subjects to make a choice between three equally likely regions of motion direction. Again, we assume uniform prior distributions of motion directions within each area. The model predicts bilateral repulsion of the estimates in the central area, leading to a strong bias that is almost independent of coherence level.

[7] Y. Weiss, E. Simoncelli, and E. Adelson. Motion illusions as optimal percept. *Nature Neuroscience*, 5(6):598–604, June 2002.

[8] A.A. Stocker and E.P. Simoncelli. Noise characteristics and prior expectations in human visual speed perception. *Nature Neuroscience*, pages 578–585, April 2006.

[9] D. Draper. Assessment and propagation of model uncertainty. *Journal of the Royal Statistical Society B*, 57:45–97, 1995.

[10] J.A. Hoeting, D. Madigan, A.E. Raftery, and C.T. Volinsky. Bayesian model averaging: A tutorial. *Statistical Science*, 14(4):382–417, 1999.

[11] T.L. Griffiths, C. Kemp, and J. Tenenbaum. *Handbook of Computational Cognitive Modeling*, chapter Bayesian models of cognition. Cambridge University Press, to appear.

[12] J.A. Yu and P. Dayan. Uncertainty, neuromodulation, and attention. *Neuron*, 46:681ff, May 2005.

[13] D. Knill. Robust cue integration: A Bayesian model and evidence from cue-conflict studies with stereoscopic and figure cues to slant. *Journal of Vision*, 7(7):1–24, May 2007.

[14] K. Körding and J. Tenenbaum. Causal inference in sensorimotor integration. In B. Schölkopf, J. Platt, and T. Hoffman, editors, *Advances in Neural Information Processing Systems 19*. MIT Press, 2007.

[15] W.J. Ma, J.M. Beck, P.E. Latham, and A. Pouget. Bayesian inference with probabilistic population codes. *Nature Neuroscience*, 9:1432ff, November 2006.

[16] Roitman J. D. Ditterich J. Mazurek, M. E. and M. N. Shadlen. A role for neural integrators in perceptual decision-making. *Cerebral Cortex*, 13:1257–1269, 2003.

[17] L. Festinger. *Theory of Cognitive Dissonance*. Stanford University Press, Stanford, CA, 1957.

[18] J.W. Brehm. Post-decision changes in the desirability of alternatives. *Journal of Abnormal and Social Psychology*, 52(3):384ff., 1956.
